# ALGONQUIN – Learning dynamic noise models from noisy speech for robust speech recognition

**Brendan J. Frey[1], Trausti T. Kristjansson[1], Li Deng[2], Alex Acero[2]**

[1] Probabilistic and Statistical Inference Group, University of Toronto
http://www.psi.toronto.edu
[2] Speech Technology Group, Microsoft Research

## Abstract

A challenging, unsolved problem in the speech recognition community is recognizing speech signals that are corrupted by loud, highly nonstationary noise. One approach to noisy speech recognition is to automatically remove the noise from the cepstrum sequence before feeding it in to a clean speech recognizer. In previous work published in *Eurospeech*, we showed how a probability model trained on clean speech and a separate probability model trained on noise could be combined for the purpose of estimating the noise-free speech from the noisy speech. We showed how an iterative 2nd order vector Taylor series approximation could be used for probabilistic inference in this model. In many circumstances, it is not possible to obtain examples of noise without speech. Noise statistics may change significantly during an utterance, so that speech-free frames are not sufficient for estimating the noise model. In this paper, we show how the noise model can be learned even when the data contains speech. In particular, the noise model can be learned from the *test* utterance and then used to denoise the test utterance. The approximate inference technique is used as an approximate E step in a generalized EM algorithm that learns the parameters of the noise model from a test utterance. For both Wall Street Journal data with added noise samples and the Aurora benchmark, we show that the new noise adaptive technique performs as well as or significantly better than the non-adaptive algorithm, without the need for a separate training set of noise examples.

## 1 Introduction

Two main approaches to robust speech recognition include "recognizer domain approaches" (Varga and Moore 1990; Gales and Young 1996), where the acoustic recognition model is modified or retrained to recognize noisy, distorted speech, and "feature domain approaches" (Boll 1979; Deng et al. 2000; Attias et al. 2001; Frey et al. 2001), where the features of noisy, distorted speech are first denoised and then fed into a speech recognition system whose acoustic recognition model is trained on clean speech.

One advantage of the feature domain approach over the recognizer domain approach is that the speech modeling part of the denoising model can have much lower com-

plexity than the full acoustic recognition model. This can lead to a much faster overall system, since the denoising process uses probabilistic inference in a much smaller model. Also, since the complexity of the denoising model is much lower than the complexity of the recognizer, the denoising model can be adapted to new environments more easily, or a variety of denoising models can be stored and applied as needed.

We model the log-spectra of clean speech, noise, and channel impulse response function using mixtures of Gaussians. (In contrast, Attias *et al.* (2001) model autoregressive coefficients.) The relationship between these log-spectra and the log-spectrum of the noisy speech is nonlinear, leading to a posterior distribution over the clean speech that is a mixture of non-Gaussian distributions. We show how a variational technique that makes use of an iterative 2nd order vector Taylor series approximation can be used to infer the clean speech and compute sufficient statistics for a generalized EM algorithm that can learn the noise model from noisy speech.

Our method, called ALGONQUIN, improves on previous work using the vector Taylor series approximation (Moreno 1996) by modeling the variance of the noise and channel instead of using point estimates, by modeling the noise and channel as a mixture mixture model instead of a single component model, by iterating Laplace's method to track the clean speech instead of applying it once at the model centers, by accounting for the error in the nonlinear relationship between the log-spectra, and by learning the noise model from noisy speech.

## 2 ALGONQUIN's Probability Model

For clarity, we present a version of ALGONQUIN that treats frames of log-spectra independently. The extension of the version presented here to HMM models of speech, noise and channel distortion is analogous to the extension of a mixture of Gaussians to an HMM with Gaussian outputs.

Following (Moreno 1996), we derive an approximate relationship between the log spectra of the clean speech, noise, channel and noisy speech. Assuming additive noise and linear channel distortion, the windowed FFT $Y(f)$ for a particular frame (25 ms duration, spaced at 10 ms intervals) of noisy speech is related to the FFTs of the channel $H(f)$, clean speech $S(f)$ and additive noise $N(f)$ by

$$Y(f) = H(f)S(f) + N(f). \tag{1}$$

We use a mel-frequency scale, in which case this relationship is only approximate. However, it is quite accurate if the channel frequency response is roughly constant across each mel-frequency filter band.

For brevity, we will assume $H(f) = 1$ in the remainder of this paper. Assuming there is no channel distortion simplifies the description of the algorithm. To see how channel distortion can be accounted for in a nonadaptive way, see (Frey et al. 2001). The technique described in this paper for adapting the noise model can be extended to adapting the channel model.

Assuming $H(f) = 1$, the energy spectrum is obtained as follows:

$$|Y(f)|^2 = Y(f)^*Y(f) = S(f)^*S(f) + N(f)^*N(f) + 2\text{Re}(N(f)^*S(f))$$
$$= |S(f)|^2 + |N(f)|^2 + 2\text{Re}(N(f)^*S(f)),$$

where "*" denotes complex conjugate. If the phase of the noise and the speech are uncorrelated, the last term in the above expression is small and we can approximate

the energy spectrum as follows:

$$|Y(f)|^2 \approx |S(f)|^2 + |N(f)|^2. \tag{2}$$

Although we could model these spectra directly, they are constrained to be non-negative. To make density modeling easier, we model the log-spectrum instead. An additional benefit to this approach is that channel distortion is an additive effect in the log-spectrum domain.

Letting $\mathbf{y}$ be the vector containing the log-spectrum $\log |Y(:)|^2$, and similarly for $\mathbf{s}$ and $\mathbf{n}$, we can rewrite (2) as

$$\exp(\mathbf{y}) \approx \exp(\mathbf{s}) + \exp(\mathbf{n}) = \exp(\mathbf{s}) \circ (\mathbf{1} + \exp(\mathbf{n} - \mathbf{s})),$$

where the exp() function operates in an element-wise fashion on its vector argument and the "∘" symbol indicates element-wise product.

Taking the logarithm, we obtain a function $\mathbf{g}()$ that is an approximate mapping of $\mathbf{s}$ and $\mathbf{n}$ to $\mathbf{y}$ (see (Moreno 1996) for more details):

$$\mathbf{y} \approx \mathbf{g}\left(\begin{bmatrix} \mathbf{s} \\ \mathbf{n} \end{bmatrix}\right) = \mathbf{s} + \ln(1 + \exp(\mathbf{n} - \mathbf{s})). \tag{4}$$

"$\mathsf{T}$" indicates matrix transpose and ln() and exp() operate on the individual elements of their vector arguments.

Assuming the errors in the above approximation are Gaussian, the observation likelihood is

$$p(\mathbf{y}|\mathbf{s}, \mathbf{n}) = \mathcal{N}\left(\mathbf{y}; \mathbf{g}\left(\begin{bmatrix} \mathbf{s} \\ \mathbf{n} \end{bmatrix}\right), \boldsymbol{\Psi}\right), \tag{5}$$

where $\boldsymbol{\Psi}$ is the diagonal covariance matrix of the errors. A more precise approximation to the observation likelihood can be obtained by writing $\boldsymbol{\Psi}$ as a function of $\mathbf{s}$ and $\mathbf{n}$, but we assume $\boldsymbol{\Psi}$ is constant for clarity.

Using a prior $p(\mathbf{s}, \mathbf{n})$, the goal of denoising is to infer the log-spectrum of the clean speech $\mathbf{s}$, given the log-spectrum of the noisy speech $\mathbf{y}$. The minimum squared error estimate of $\mathbf{s}$ is $\hat{\mathbf{s}} = \int_{\mathbf{s}} \mathbf{s} p(\mathbf{s}|\mathbf{y})$, where $p(\mathbf{s}|\mathbf{y}) \propto \int_{\mathbf{n}} p(\mathbf{y}|\mathbf{s}, \mathbf{n}) p(\mathbf{s}, \mathbf{n})$. This inference is made difficult by the fact that the nonlinearity $\mathbf{g}([\mathbf{s}\ \mathbf{n}]^{\mathsf{T}})$ in (5) makes the posterior non-Gaussian even if the prior is Gaussian. In the next section, we show how an iterative variational method that uses a 2nd order vector Taylor series approximation can be used for approximate inference and learning.

We assume that *a priori* the speech and noise are independent — $p(\mathbf{s}, \mathbf{n}) = p(\mathbf{s})p(\mathbf{n})$ — and we model each using a separate mixture of Gaussians. $c^s = 1, \dots, N^s$ is the class index for the clean speech and $c^n = 1, \dots, N^n$ is the class index for the noise. The mixing proportions and Gaussian components are parameterized as follows:

$$p(\mathbf{s}) = \sum_{c^s} p(c^s)p(\mathbf{s}|c^s), \quad p(c^s) = \pi_{c^s}^s, \quad p(\mathbf{s}|c^s) = \mathcal{N}(\mathbf{s}; \boldsymbol{\mu}_{c^s}^s, \boldsymbol{\Sigma}_{c^s}^s),$$

$$p(\mathbf{n}) = \sum_{c^n} p(c^n)p(\mathbf{n}|c^n), \quad p(c^n) = \pi_{c^n}^n, \quad p(\mathbf{n}|c^n) = \mathcal{N}(\mathbf{n}; \boldsymbol{\mu}_{c^n}^n, \boldsymbol{\Sigma}_{c^n}^n). \tag{6}$$

We assume the covariance matrices $\boldsymbol{\Sigma}_{c^s}^s$ and $\boldsymbol{\Sigma}_{c^n}^n$ are diagonal.

Combining (5) and (6), the joint distribution over the noisy speech, clean speech class, clean speech vector, noise class and noise vector is

$$p(\mathbf{y}, \mathbf{s}, c^s, \mathbf{n}, c^n) = \mathcal{N}\left(\mathbf{y}; \mathbf{g}\left(\begin{bmatrix} \mathbf{s} \\ \mathbf{n} \end{bmatrix}\right), \boldsymbol{\Psi}\right) \pi_{c^s}^s \mathcal{N}(\mathbf{s}; \boldsymbol{\mu}_{c^s}^s, \boldsymbol{\Sigma}_{c^s}^s) \pi_{c^n}^n \mathcal{N}(\mathbf{n}; \boldsymbol{\mu}_{c^n}^n, \boldsymbol{\Sigma}_{c^n}^n). \tag{7}$$

Under this joint distribution, the posterior $p(\mathbf{s}, \mathbf{n}|\mathbf{y})$ is a mixture of non-Gaussian distributions. In fact, for a given speech class and noise class, the posterior $p(\mathbf{s}, \mathbf{n}|c^s, c^n, \mathbf{y})$ may have multiple modes. So, exact computation of $\hat{\mathbf{s}}$ is intractable and we use an approximation.

# 3   Approximating the Posterior

For the current frame of noisy speech $\mathbf{y}$, ALGONQUIN approximates the posterior using a simpler, parameterized distribution, $q$:

$$p(\mathbf{s}, c^s, \mathbf{n}, c^n | \mathbf{y}) \approx q(\mathbf{s}, c^s, \mathbf{n}, c^n).$$

The "variational parameters" of $q$ are adjusted to make this approximation accurate, and then $q$ is used as a surrogate for the true posterior when computing $\hat{\mathbf{s}}$ and learning the noise model (c.f. (Jordan et al. 1998)).

For each $c^s$ and $c^n$, we approximate $p(\mathbf{s}, \mathbf{n} | c^s, c^n, \mathbf{y})$ by a Gaussian,

$$q(\mathbf{s}, \mathbf{n} | c^s, c^n) = \mathcal{N}\left( \begin{bmatrix} \mathbf{s} \\ \mathbf{n} \end{bmatrix}; \begin{bmatrix} \boldsymbol{\eta}^s_{c^s c^n} \\ \boldsymbol{\eta}^n_{c^s c^n} \end{bmatrix}, \begin{bmatrix} \boldsymbol{\Phi}^{ss}_{c^s c^n} & \boldsymbol{\Phi}^{sn}_{c^s c^n} \\ \boldsymbol{\Phi}^{ss}_{c^s c^n} & \boldsymbol{\Phi}^{nn}_{c^s c^n} \end{bmatrix} \right), \tag{9}$$

where $\boldsymbol{\eta}^s_{c^s c^n}$ and $\boldsymbol{\eta}^n_{c^s c^n}$ are the approximate posterior means of the speech and noise for classes $c^s$ and $c^n$, and $\boldsymbol{\Phi}^{ss}_{c^s c^n}$, $\boldsymbol{\Phi}^{nn}_{c^s c^n}$ and $\boldsymbol{\Phi}^{sn}_{c^s c^n}$ specify the covariance matrix for the speech and noise for classes $c^s$ and $c^n$. Since rows of vectors in (4) do not interact and since the likelihood covariance matrix $\boldsymbol{\Psi}$ and the prior covariance matrices $\boldsymbol{\Sigma}^s_{c^s}$ and $\boldsymbol{\Sigma}^n_{c^n}$ are diagonal, the matrices $\boldsymbol{\Phi}^{ss}_{c^s c^n}$, $\boldsymbol{\Phi}^{nn}_{c^s c^n}$ and $\boldsymbol{\Phi}^{sn}_{c^s c^n}$ are diagonal.

The posterior mixing proportions for classes $c^s$ and $c^n$ are $q(c^s, c^n) = \rho_{c^s c^n}$. The approximate posterior is given by $q(\mathbf{s}, \mathbf{n}, c^s, c^n) = q(\mathbf{s}, \mathbf{n} | c^s, c^n) q(c^s, c^n)$.

The goal of variational inference is to minimize the relative entropy (Kullback-Leibler divergence) between $q$ and $p$:

$$\mathcal{K} = \sum_{c^s} \sum_{c^n} \int_{\mathbf{s}} \int_{\mathbf{n}} q(\mathbf{s}, \mathbf{n}, c^s, c^n) \ln \frac{q(\mathbf{s}, \mathbf{n}, c^s, c^n)}{p(\mathbf{s}, c^s, \mathbf{n}, c^n | \mathbf{y})}.$$

This is a particularly good choice for a cost function, because, since $\ln p(\mathbf{y})$ doesn't depend on the variational parameters, minimizing $\mathcal{K}$ is equivalent to maximizing

$$\mathcal{F} = \ln p(\mathbf{y}) - \mathcal{K} = \sum_{c^s} \sum_{c^n} \int_{\mathbf{s}} \int_{\mathbf{n}} q(\mathbf{s}, \mathbf{n}, c^s, c^n) \ln \frac{p(\mathbf{s}, c^s, \mathbf{n}, c^n, \mathbf{y})}{q(\mathbf{s}, \mathbf{n}, c^s, c^n)},$$

which is a lower bound on the log-probability of the data. So, variational inference can be used as a generalized E step (Neal and Hinton 1998) in an algorithm that alternatively maximizes a lower bound on $\ln p(\mathbf{y})$ with respect to the variational parameters and the noise model parameters, as described in the next section.

Variational inference begins by optimizing the means and variances in (9) for each $c^s$ and $c^n$. Initially, we set the posterior means and variances to the prior means and variances. $\mathcal{F}$ does not have a simple form in these variational parameters. So, at each iteration, we make a 2nd order vector Taylor series approximation of the likelihood, centered at the current variational parameters, and maximize the resulting approximation to $\mathcal{F}$. The updates are

$$\begin{bmatrix} \boldsymbol{\Phi}^{ss}_{c^s c^n} & \boldsymbol{\Phi}^{sn}_{c^s c^n} \\ \boldsymbol{\Phi}^{ss}_{c^s c^n} & \boldsymbol{\Phi}^{nn}_{c^s c^n} \end{bmatrix} \leftarrow \left( \begin{bmatrix} \boldsymbol{\Sigma}^s_{c^s} & \mathbf{0} \\ \mathbf{0} & \boldsymbol{\Sigma}^n_{c^n} \end{bmatrix}^{-1} + \mathbf{g}'\left( \begin{bmatrix} \boldsymbol{\eta}^s_{c^s c^n} \\ \boldsymbol{\eta}^n_{c^s c^n} \end{bmatrix} \right)^{\mathsf{T}} \boldsymbol{\Psi}^{-1} \mathbf{g}'\left( \begin{bmatrix} \boldsymbol{\eta}^s_{c^s c^n} \\ \boldsymbol{\eta}^n_{c^s c^n} \end{bmatrix} \right) \right)^{-1}, \quad \text{and}$$

$$\begin{pmatrix} \boldsymbol{\eta}^s_{c^s c^n} \\ \boldsymbol{\eta}^n_{c^s c^n} \end{pmatrix} \leftarrow \begin{pmatrix} \boldsymbol{\eta}^s_{c^s c^n} \\ \boldsymbol{\eta}^n_{c^s c^n} \end{pmatrix} +$$

$$\begin{bmatrix} \boldsymbol{\Phi}^{ss}_{c^s c^n} & \boldsymbol{\Phi}^{sn}_{c^s c^n} \\ \boldsymbol{\Phi}^{ss}_{c^s c^n} & \boldsymbol{\Phi}^{nn}_{c^s c^n} \end{bmatrix} \left( \begin{bmatrix} \boldsymbol{\Sigma}^s_{c^s} & \mathbf{0} \\ \mathbf{0} & \boldsymbol{\Sigma}^n_{c^n} \end{bmatrix}^{-1} \begin{bmatrix} \boldsymbol{\mu}^s_{c^s} - \boldsymbol{\eta}^s_{c^s c^n} \\ \boldsymbol{\mu}^n_{c^n} - \boldsymbol{\eta}^n_{c^s c^n} \end{bmatrix} + \mathbf{g}'\left( \begin{bmatrix} \boldsymbol{\eta}^s_{c^s c^n} \\ \boldsymbol{\eta}^n_{c^s c^n} \end{bmatrix} \right)^{\mathsf{T}} \boldsymbol{\Psi}^{-1} \left( \mathbf{y} - \mathbf{g}\left( \begin{bmatrix} \boldsymbol{\eta}^s_{c^s c^n} \\ \boldsymbol{\eta}^n_{c^s c^n} \end{bmatrix} \right) \right) \right),$$

where $\mathbf{g}'()$ is a matrix of derivatives whose rows correspond to the noisy speech $\mathbf{y}$ and whose columns correspond to the clean speech and noise $[\mathbf{s} \ \mathbf{n}]$.

The inverse posterior covariance matrix is the sum of the inverse prior covariance matrix and the inverse likelihood covariance matrix, modified by the Jacobian $\mathbf{g}'()$ for the mapping from $\mathbf{s}$ and $\mathbf{n}$ to $\mathbf{y}$

The posterior means are moved towards the prior means and toward values that match the observation $\mathbf{y}$. These two effects are weighted by the inverse prior covariance matrix and the inverse likelihood covariance matrix.

After iterating the above updates (in our experiments, 3 to 5 times) for each $c^s$ and $c^n$, the posterior mixing proportions that maximize $\mathcal{F}$ are computed:

$$\rho_{c^s c^n} = \lambda \pi_{c^s}^s \pi_{c^n}^n \exp\left(-\frac{1}{2}\ln|\mathbf{\Sigma}_{c^s}^s| - \frac{1}{2}\ln|\mathbf{\Sigma}_{c^n}^n| + \frac{1}{2}\ln\left|\begin{matrix}\mathbf{\Phi}_{c^s c^n}^{ss} & \mathbf{\Phi}_{c^s c^n}^{sn}\\ \mathbf{\Phi}_{c^s c^n}^{ss} & \mathbf{\Phi}_{c^s c^n}^{nn}\end{matrix}\right|\right.$$

$$-\frac{1}{2}\left(\mathbf{y}-\mathbf{g}\left(\begin{bmatrix}\boldsymbol{\eta}_{c^s c^n}^s\\ \boldsymbol{\eta}_{c^s c^n}^n\end{bmatrix}\right)\right)^{\mathsf{T}}\mathbf{\Psi}^{-1}\left(\mathbf{y}-\mathbf{g}\left(\begin{bmatrix}\boldsymbol{\eta}_{c^s c^n}^s\\ \boldsymbol{\eta}_{c^s c^n}^n\end{bmatrix}\right)\right) - \frac{1}{2}\mathrm{tr}\left(\mathbf{g}\left(\begin{bmatrix}\boldsymbol{\eta}_{c^s c^n}^s\\ \boldsymbol{\eta}_{c^s c^n}^n\end{bmatrix}\right)^{\mathsf{T}}\mathbf{\Psi}^{-1}\mathbf{g}\left(\begin{bmatrix}\boldsymbol{\eta}_{c^s c^n}^s\\ \boldsymbol{\eta}_{c^s c^n}^n\end{bmatrix}\right)\begin{bmatrix}\mathbf{\Phi}_{c^s c^n}^{ss} & \mathbf{\Phi}_{c^s c^n}^{sn}\\ \mathbf{\Phi}_{c^s c^n}^{ss} & \mathbf{\Phi}_{c^s c^n}^{nn}\end{bmatrix}\right)$$

$$-\frac{1}{2}(\boldsymbol{\eta}_{c^s c^n}^s - \boldsymbol{\mu}_{c^s}^s)^{\mathsf{T}}\mathbf{\Sigma}_{c^s}^{s-1}(\boldsymbol{\eta}_{c^s c^n}^s - \boldsymbol{\mu}_{c^s}^s) - \frac{1}{2}\mathrm{tr}(\mathbf{\Sigma}_{c^s}^{s-1}\mathbf{\Phi}_{c^s c^n}^{ss})$$

$$\left.-\frac{1}{2}(\boldsymbol{\eta}_{c^s c^n}^n - \boldsymbol{\mu}_{c^n}^n)^{\mathsf{T}}\mathbf{\Sigma}_{c^n}^{n-1}(\boldsymbol{\eta}_{c^s c^n}^n - \boldsymbol{\mu}_{c^n}^n) - \frac{1}{2}\mathrm{tr}(\mathbf{\Sigma}_{c^n}^{n-1}\mathbf{\Phi}_{c^s c^n}^{nn})\right),$$

where $\lambda$ is a normalizing constant that is computed so that $\sum_{c^s c^n} \rho_{c^s c^n} = 1$. The minimum squared error estimate of the clean speech, $\hat{\mathbf{s}}$, is

$$\hat{\mathbf{s}} = \sum_{c^s c^n} \rho_{c^s c^n} \boldsymbol{\eta}_{c^s c^n}^s.$$

We apply this algorithm on a frame-by-frame basis, until all frames in the test utterance have been denoised.

## 4 Speed

Since elements of $\mathbf{s}$, $\mathbf{n}$ and $\mathbf{y}$ that are in different rows *do not* interact in (4), the above matrix algebra reduces to efficient scalar algebra. For 256 speech components, 4 noise components and 3 iterations of inference, our unoptimized C code takes 60 ms to denoise each frame. We are confident that this time can be reduced by an order of magnitude using standard implementation tricks.

## 5 Adapting the Noise Model Using Noisy Speech

The version of ALGONQUIN described above requires that a mixture model of the noise be trained on noise samples, before the log-spectrum of the noisy speech can be denoised. Here, we describe how the iterative inference technique can be used as the E step in a generalized EM algorithm for learning the noise model from noisy speech.

For a set of frames $\mathbf{y}^{(1)}, \ldots, \mathbf{y}^{(T)}$ in a noisy test utterance, we construct a total bound

$$\mathcal{F} = \sum_t \mathcal{F}^{(t)} \leq \sum_t \ln p(\mathbf{y}^{(t)}).$$

The generalized EM algorithm alternates between updating one set of variational parameters $\rho_{c^s c^n}^{(t)}$, $\boldsymbol{\eta}_{c^s c^n}^{n(t)}$, *etc.* for each frame $t = 1, \ldots, T$, and maximizing $\mathcal{F}$ with respect to the noise model parameters $\pi_{c^n}^n$, $\boldsymbol{\mu}_{c^n}^n$ and $\mathbf{\Sigma}_{c^n}^n$. Since $\mathcal{F} \leq \sum_t \ln p(\mathbf{y}^{(t)})$, this procedure maximizes a lower bound on the log-probability of the data. The use of the vector Taylor series approximations leads to an algorithm that maximizes an approximation to a lower bound on the log-probability of the data.

|          | Restaurant | Street | Airport | Station | Average |
|----------|-----------:|-------:|--------:|--------:|--------:|
| 20 dB    | 2.12       | 2.96   | 1.82    | 1.73    | 2.16    |
| 15 dB    | 3.87       | 4.78   | 2.27    | 3.24    | 3.54    |
| 10 dB    | 9.18       | 10.73  | 5.49    | 6.48    | 7.97    |
| 5 dB     | 20.51      | 13.52  | 14.97   | 15.18   | 18.54   |
| 0 dB     | 47.04      | 45.68  | 36.00   | 37.24   | 41.49   |
| -5dB     | 78.69      | 72.34  | 69.04   | 67.26   | 71.83   |
| Average  | 16.54      | 17.53  | 12.11   | 12.77   | **14.74** |

Table 1: Word error rates (in percent) on set B of the Aurora test set, for the adaptive version of ALGONQUIN with 4 noise componentsset.

Setting the derivatives of $\mathcal{F}$ with respect to the noise model parameters to zero, we obtain the following M step updates:

$$\pi^n_{c^n} \leftarrow \frac{1}{T} \sum_t \sum_{c^s} \rho^{(t)}_{c^s c^n},$$

$$\boldsymbol{\mu}^n_{c^n} \leftarrow \left( \sum_t \sum_{c^s} \rho^{(t)}_{c^s c^n} \boldsymbol{\eta}^{n\,(t)}_{c^s c^n} \right) \Big/ \left( \sum_t \sum_{c^s} \rho^{(t)}_{c^s c^n} \right),$$

$$\boldsymbol{\Sigma}^n_{c^n} \leftarrow \left( \sum_t \sum_{c^s} \rho^{(t)}_{c^s c^n} \left( \boldsymbol{\Phi}^{nn(t)}_{c^s c^n} + \mathrm{diag}\left( (\boldsymbol{\eta}^{n\,(t)}_{c^s c^n} - \boldsymbol{\mu}^n_{c^n})(\boldsymbol{\eta}^{n\,(t)}_{c^s c^n} - \boldsymbol{\mu}^n_{c^n})^\top \right) \right) \right) \Big/ \left( \sum_t \sum_{c^s} \rho^{(t)}_{c^s c^n} \right).$$

The variational parameters can be updated multiple times before updating the model parameters, or the variational parameters can updated only once before updating the model parameters. The latter approach may converge more quickly in some situations.

## 6 Experimental Results

After training a 256-component speech model on clean speech, we used the adaptive version of ALGONQUIN to denoise noisy test utterances on two tasks: the publically available Aurora limited vocabulary speech recognition task (http://www.etsi.org/technicalactiv/dsr.htm); the Wall Street Journal (WSJ) large vocabulary speech recognition task, with Microsoft's Whisper speech recognition system.

We obtained results on all 48 test sets from partitions A and B of the Aurora database. Each set contains 24,000 sentences that have been corrupted from one of 4 different noise types and one of 6 different signal to noise ratios. Table 1 gives the error rates for the adaptive version of ALGONQUIN, with 4 noise components. These error rates are superior to error rates obtained by our spectral subtraction technique for (Deng et al. 2000), and highly competitive with other results on the Aurora task.

Table 2 compares the performances of the adaptive version of ALGONQUIN and the non-adaptive version. For the non-adaptive version, 20 non-speech frames are used to estimate the noise model. For the adaptive version, the parameters are initialized using 20 non-speech frames and then 3 iterations of generalized EM are used to learn the noise model. The average error rate over all noise types and SNRs for set B of Aurora drops from 17.65% to 15.19% when the noise adaptive algorithm is used to update the noise model. This is a relative gain of 13.94%. When 4 components are used there is a further gain of 2.5%.

The Wall Street Journal test set consists of 167 sentences spoken by female speakers. The Microsoft Whisper recognizer with a 5,000 word vocabulary was used to recognize these sentences. Table 2 shows that the adaptive version of algonquin

| | WER 20 frames | WER 1 comp | Reduction in WER | WER 4 comps | Reduction in WER |
|---|---|---|---|---|---|
| Aurora, Set A | 18.10% | 15.91% | 12.10% | 15.62% | 13.70% |
| Aurora, Set B | 17.65% | 15.19% | 13.94% | 14.74% | 16.49% |
| WSJ, XD14, 10dB | 30.00% | 21.8% | 27.33% | 21.50% | 28.33% |
| WSJ, XD10, 10dB | 21.80% | 20.6% | 5.50% | 20.6% | 5.50 % |

Table 2: Word error rates (WER) and percentage reduction in WER for the Aurora test data and the Wall Street Journal test data, *without scaling.*

performs better than the non-adaptive version, especially on noise type "XD14", which consists of the highly-nonstationary sound of a jet engine shutting down. For noise type "XD10", which is stationary noise, we observe a gain, but we do not see any further gain for multiple noise components.

## 7 Conclusions

A far as variational methods go, ALGONQUIN is a fast technique for denoising log-spectrum or cepstrum speech feature vectors. ALGONQUIN improves on previous work using the vector Taylor series approximation, by using multiple component speech and noise models, and it uses an iterative variational method to produce accurate posterior distributions for speech and noise. By employing a generalized EM method, ALGONQUIN can estimate a noise model from noisy speech data.

Our results show that the noise adaptive ALGONQUIN algorithm can obtain better results than the non-adaptive version. This is especially important for non-stationary noise, where the non-adaptive algorithm relies on an estimate of the noise based on a subset of the frames, but the adaptive algorithm uses all the frames of the utterance, even those that contain speech.

A different approach to denoising speech features is to learn time-domain models. Attias *et al.* (2001) report results on a *non-adaptive* time-domain technique. Our results cannot be directly compared with theirs, since our results are for unscaled data. Eventually, the two approaches should be thoroughly compared.

## References

Attias, H., Platt, J. C., Acero, A., and Deng, L. 2001. Speech denoising and dereverberation using probabilistic models. In *Advances in Neural Information Processing Systems 13*. MIT Press, Cambridge MA.

Boll, S. 1979. Suppression of acoustic noise in speech using spectral subtraction. *IEEE Transactions on Acoustics, Speech and Signal Processing*, 27:114–120.

Deng, L., Acero, A., Plumpe, M., and Huang, X. D. 2000. Large-vocabulary speech recognition under adverse acoustic environments. In *Proceedings of the International Conference on Spoken Language Processing*, pages 806–809.

Frey, B. J., Deng, L., Acero, A., and Kristjansson, T. 2001. ALGONQUIN: Iterating Laplace's method to remove multiple types of acoustic distortion for robust speech recognition. In *Proceedings of Eurospeech 2001*.

Gales, M. J. F. and Young, S. J. 1996. Robust continuous speech recognition using parallel model combination. *IEEE Speech and Audio Processing*, 4(5):352–359.

Jordan, M. I., Ghahramani, Z., Jaakkola, T. S., and Saul, L. K. 1998. An introduction to variational methods for graphical models. In Jordan, M. I., editor, *Learning in Graphical Models*. Kluwer Academic Publishers, Norwell MA.

Moreno, P. 1996. *Speech Recognition in Noisy Environments*. Carnegie Mellon University, Pittsburgh PA. Doctoral dissertation.

Neal, R. M. and Hinton, G. E. 1998. A view of the EM algorithm that justifies incremental, sparse, and other variants. In Jordan, M. I., editor, *Learning in Graphical Models*, pages 355–368. Kluwer Academic Publishers, Norwell MA.

Varga, A. P. and Moore, R. K. 1990. Hidden Markov model decomposition of speech and noise. In *Proceedings of the International Conference on Acoustics, Speech and Signal Processing*, pages 845–848. IEEE Press.
